# An Additive Latent Feature Model for Transparent Object Recognition

**Mario Fritz**
UC Berkeley

**Michael Black**
Brown University

**Gary Bradski**
Willow Garage

**Sergey Karayev**
UC Berkeley

**Trevor Darrell**
UC Berkeley

## Abstract

Existing methods for visual recognition based on quantized local features can perform poorly when local features exist on transparent surfaces, such as glass or plastic objects. There are characteristic patterns to the local appearance of transparent objects, but they may not be well captured by distances to individual examples or by a local pattern codebook obtained by vector quantization. The appearance of a transparent patch is determined in part by the refraction of a background pattern through a transparent medium: the energy from the background usually dominates the patch appearance. We model transparent local patch appearance using an additive model of latent factors: background factors due to scene content, and factors which capture a local edge energy distribution characteristic of the refraction. We implement our method using a novel LDA-SIFT formulation which performs LDA prior to any vector quantization step; we discover latent topics which are characteristic of particular transparent patches and quantize the SIFT space into transparent visual words according to the latent topic dimensions. No knowledge of the background scene is required at test time; we show examples recognizing transparent glasses in a domestic environment.

## 1 Introduction

Household scenes commonly contain transparent objects such as glasses and bottles made of various materials (like those in Fig. 6). Instance and category recognition of such objects is important for applications including domestic service robotics and image search. Despite the prevalence of transparent objects in human environments, the problem of transparent object recognition has received relatively little attention. We argue that current appearance-based methods for object and category recognition are not appropriate for transparent objects where the appearance can change dramatically depending on the background and illumination conditions. A full physically plausible generative model of transparent objects is currently impractical for recognition tasks. Instead we propose a new latent component representation that allows us to learn *transparent visual words* that capture locally discriminative visual features on transparent objects.

Figure 1 shows an example of a transparent object observed in front of several different background patterns; the local edge energy histogram is shown around a fixed point on the object for each image. While the overall energy pattern is quite distinct, there is a common structure that can be observed across each observation. This structure can be estimated from training examples and detected reliably in test images: we form a local statistical model of transparent patch appearance by estimating a latent local factor model from training data which includes varying background imagery. The varying background provides examples of how the transparent objects refracts light,

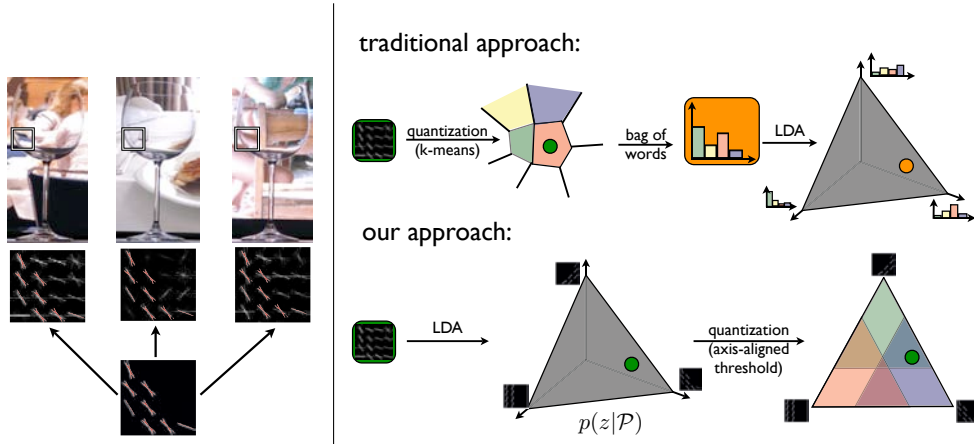

Figure 1: Left: Images of a transparent object in different environments. A point on the object is highlighted in each image, and the local orientation edge energy map is shown. While the background dominates the local patch, there is a latent structure that is discriminative of the object. Right: Our model finds local transparent structure by applying a latent factor model (e.g., LDA) before a quantization step. In contrast to previous approaches that applied such models to a quantized visual word model, we apply them directly to the SIFT representation, and then quantize the resulting model into descriptors according to the learned topic distribution.

an idea has been used as a way of capturing the refractive properties of glass [34] but not, to our knowledge, as a way of training an object recognition system.

Specifically, we adopt a hybrid generative-discriminative model in the spirit of [13] in which a generative latent factor model discovers a vocabulary of locally transparent patterns, and a discriminant classifier is applied to the space of these activations to detect a category of interest. Our latent component representation decomposes patch appearance into sub-components based on an additive model of local patch formation; in particular we use the latent Dirichlet allocation (LDA) model in our experiments below. Transparent object recognition is achieved using a simple probabilistic model of likely local object features. A latent topic model is learned over the space of local patches in images of a given object observed with varying backgrounds; clustering in this space yields descriptors that can be used to infer transparent structures in an image at test time without any knowledge of the underlying background pattern or environmental illumination. Each image patch at test time is then labeled with one or more candidate quantized latent structures (topics), which define our transparent visual word identifiers.

Currently, the study of transparent object recognition is extremely limited and we believe ours is the first to consider category recognition of transparent objects in natural settings, with varying pose and unconstrained illumination. The paper provides a first exploration of the problem, establishes a baseline, demonstrates feasibility and suggests problems for future work. Our results show that recognition of transparent objects is possible without explicit physically-based refraction and reflection models, using a learning-based additive latent local feature appearance model.

## 2 Related Work

There is an extensive literature of local feature detection and description techniques; here we focus on those related to our transparent object recognition formulation. Existing methods for object category and object instance recognition are generally designed for opaque objects, typically finding characteristic local patches using descriptors based on weighted histograms of local orientation energy [2, 18, 6], locally stable region characteristics [19], local self-similarity [29], etc.

We explore a similar direction but extend this work to transparent objects. Specifically, we base our method on a novel combination of SIFT [18] and latent Dirichlet allocation (LDA) [4], two techniques used in many previous object recognition methods. The SIFT descriptor (see also the related HOG [6] and neurally plausible HMAX models [27]) generally characterizes local appearance

with a spatial grid of histograms, with each histogram aggregating a number of edges at a particular orientation in a grid cell.

Approaches based on quantizing or matching local appearance from single observations can perform poorly on objects that are made of transparent material. The local appearance of a transparent object is governed, in general, by a complex rendering process including multi-layer refraction and specular reflections. The local appearance of a particular point on a transparent object may be dominated by environmental characteristics, i.e., the background pattern and illumination field. Models that search for nearest neighbor local appearance patterns from training instances may identify the environment (e.g. the background behind the object) rather than the object of interest. Methods that vector quantize individual observations of local appearance may learn a representation that partitions well the variation in the environment. Neither approach is likely to learn salient characteristics of local transparent appearance.

Bag-of-words (c.f., [31, 5, 24], and many others), Pyramid-match [14, 17], and many generative methods [11, 32] exploit the "visual word" metaphor, establishing a vocabulary of quantized SIFT appearance. Typically a k-means clustering method (or a discriminative variant) is used to associate nearby appearances into a single cluster. Unfortunately when background energy dominates transparent foreground energy, averaging similar local appearances may simply find a cluster center corresponding to background structure, not foreground appearance.

For transparent objects, we argue that there is local latent structure that can be used to recognize objects; we formulate the problem as learning this structure in a SIFT representation using a latent factor model. Early methods for probabilistic topic modeling (e.g. [16]) were developed in the domain of text analysis to factor word occurrence distributions of documents in to multiple latent topics in an unsupervised manner. Latent Dirichlet Allocation [4, 15] is an additive topic model, that allows for prior distributions on mixing proportions as well as the components.

SIFT and LDA have been combined before, but the conventional application of LDA to SIFT is to form a topic representation over the quantized SIFT descriptors [30, 32, 10, 22]. As previous methods apply vector quantization before latent modeling, they are inappropriate for uncovering latent (and possibly subtle) transparent structures. To our knowledge, ours is the first work to infer a latent topic model from a SIFT representation *before* quantizing into a "visual word" representation.

Related work on latent variable models includes [9], which reports a "LatentSVM" model to solve for a HOG descriptor with enumerated local high resolution patches. The offset of the patches is regarded as a latent variable in the method, and is solved using a semi-convex optimization. Note that the latent variable here is distinct from a latent topic variable, and that there are no explicitly shared structures across the parts in their model. Quattoni *et al.* [26] report an object recognition model that uses latent or hidden variables which have CRF-like dependencies to observed image features, including a representation that is formed with local oriented structures. Neither method has an LDA component, but both of these methods have considerable representational flexibility and the ability to learn weight factors from large amounts of training data. Our method is similar in spirit to [13], which uses local oriented gradient strength in a HOG descriptor as a word in an LDA model. However, our method is based on local patches, while theirs is evaluated over a global descriptor; their model also did not include any explicit quantization into discrete (and overlapping) visual words. No results have been reported on transparent objects using these methods.

In addition to the above work on generic (non-transparent) object recognition, there has been some limited work in the area of transparent object recognition. Most relevant is that of [25], which focuses on recognition from specular reflections. If the lighting conditions and pose of the object are known, then specularities on the glass surface can be highly discriminative of different object shapes. The initial work in [25] however assumes a highly simplified environment and has not been tested with unknown 3D shape, or with varying and unknown pose and complex illumination. By focusing on specularities they also ignore the potentially rich source of information about transparent object shape caused by the refraction of the background image structure. We take a different approach and do not explicitly model specular reflections or their relationship to 3D shape. Rather than focus on a few highlights we focus on how transparent objects appear against varied backgrounds. Our learning approach is designed to automatically uncover the most discriminative latent features in the data (which may include specular reflections).

It is important to emphasize that we are approaching this problem as one of transparent *object* recognition. This is in contrast to previous work that has explored glass *material* recognition [20, 21]. This is analogous to the distinction between modeling "things" and "stuff" [1]. There has been significant work on detecting and modeling surfaces that are specular or transparent [7, 12, 23, 28]. These methods, which focus on material recognition, may give important insight into the systematic deformations of the image statistics caused by transparent objects and may inform the design of features for object recognition. Note that a generic "glass material" detector would complement our approach in that it could focus attention on regions of a scene that are most likely to contain transparent objects. Thus, while material recognition and surface modeling are distinct from our problem, we consider them complimentary.

## 3  Local Transparent Features

Local transparent patch appearance can be understood as a combination of different processes that involve illuminants in the scene, overall 3D structure, as well as the geometry and material properties of the transparent object. Many of these phenomena can be approximated with an additive image formation model, subject to certain deformations. A full treatment of the refractive properties of different transparent materials and their geometry is beyond our scope and likely intractable for most contemporary object recognition tasks.

Rather than analytically model interactions between scene illumination, material properties and object geometry, we take a machine learning perspective and assume that observed image patches factor into latent components – some originating from the background, others reflecting the structure of the transparent object. To detect a transparent object it may be sufficient to detect characteristic patterns of deformation (e.g. in the stem of a wine glass) or features that are sometimes present in the image and sometimes not (like the rim of a thin glass).

We assume a decomposition of an image $\mathcal{I}$ into a set of densely sampled image patches $\mathcal{I}_{\mathcal{P}}$, each represented by a local set of edge responses in the style of [18, 6], which we further model with an additive process. From each $\mathcal{I}_{\mathcal{P}}$ we obtain local gradient estimates $G_{\mathcal{P}}$. We model local patch appearance as an additive combination of image structures originating from a background patch appearance $\mathcal{A}_0$ as well as a one or more patterns $\mathcal{A}_i$ that has been affected by e.g., refraction of the transparent object. An image patch is thus described by:

$$G_{\mathcal{P}} = [\, g_{\mathcal{P}}(0,0,0), \ldots, g_{\mathcal{P}}(M,N,T) \,] = \sum_i \theta^{(i)} A_i \tag{1}$$

where $g_{\mathcal{P}}(i,j,o)$ is the edge count for a histogram bin at position $(i,j)$ in patch $\mathcal{I}_{\mathcal{P}}$ at orientation index $o$; $M, N, T$ give the dimensions of the descriptor histogram and $\theta^{(i)}$ is the scalar weight associated with pattern $A_i$. We further assume non-negative $\theta^{(i)}$, reflecting the image formation process.

Based on this model, we formulate a corresponding generative process for the local gradient statistics $p(G_{\mathcal{P}})$ for patch $\mathcal{P}$. The model constitutes a decomposition of $p(G_{\mathcal{P}})$ into components $p(G|z=j)$ and mixing proportions $p(z=j)$.

$$p(G_{\mathcal{P}}) = \sum_j^T p(G|z=j)p(z=j). \tag{2}$$

Both the components as well as their mixing proportions are unknown to us wherefore we treat them as latent variables in our model. However, we may reasonably assume that each observed patch was only generated from a few components, so we employ a sparseness prior over the component weights. To estimate this mixture model we use methods for probabilistic topic modeling that allow us to place prior distributions on mixing proportions as well as the components. Based on a set of training patches, we learn a model over the patches which captures the salient structures characterizing the object patch appearance as a set of latent topics. We have investigated both supervised and unsupervised latent topic formation strategies; as reported below both outperform

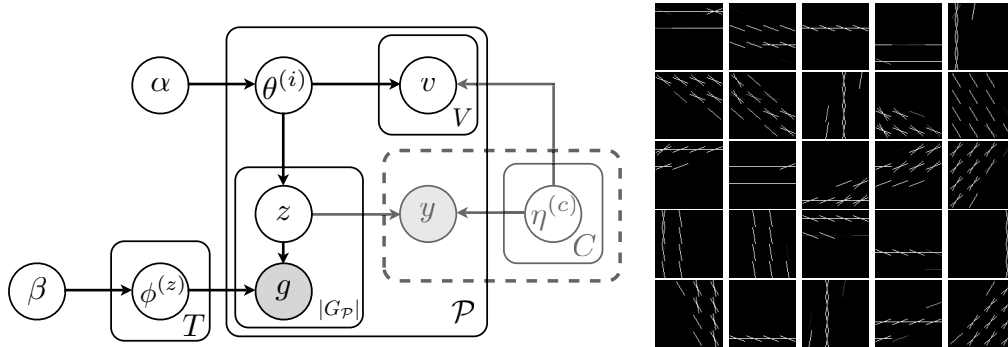

Figure 2: Left: Graphical model representing our latent topic model of patch appearance and quantization into a potentially overlapping set of visual words. See text for details. Right: Local factors learned by latent topic model for example training data.

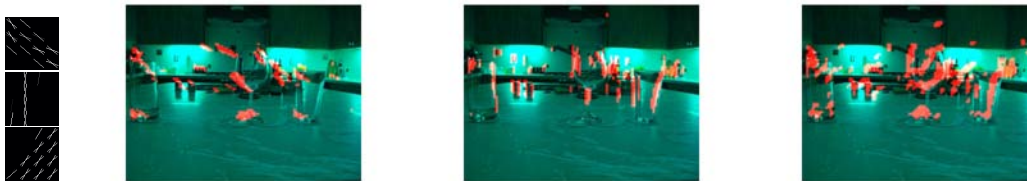

Figure 3: Detected quantized transparent local features (transparent visual words) on an example image. Each image shows the detected locations for the transparent visual word corresponding to the latent topics depicted on the left.

traditional quantized appearance techniques. Figure 2 illustrates examples of the latent topics $\phi^{(z)}$ learned by decomposing a local SIFT representation into underlying components. At test time, a patch is presented to the LDA model and topic activation weights are inferred given the fixed topic vectors.

To obtain a discrete representation, we can quantize the space of topic vectors into 'transparent visual words'. The essence of transparency is that more than one visual word may be present in a single local patch, so we have an overlapping set of clusters in the topic space. We quantize the topic activation levels $\theta^{(i)}$ into a set of overlapping visual words by forming axis-aligned partitions of the topic space and associate a distinct visual word detection with each topic activation value that is above a threshold activation level $\epsilon$.

Figure 2 summarizes our transparent visual word model in a graphical model representation. Our method follows the standard LDA presentation, with the addition of a plate of variables corresponding to visual word detections. These boolean detection variables deterministically depend on the latent topic activation vector: word $v_i$ is set when $\theta^{(i)} \geq \epsilon$. Figure 3 illustrates detected local features on an example image.

Latent topics can be found using an unsupervised process, where topics are trained from a generic corpus of foreground and/or background imagery. More discriminative latent factors can be found by taking advantage of supervised patch labels. In this case we employ a supervised extension to the LDA model[1] (sLDA [3]), which allows us to provide the model with class labels per patch in order to train a discriminant representation. This revised model is displayed in the dashed box in Figure 2. The foreground/background label for each patch is provided at training time by the observed variable $y$; the parameters $\eta^{(c)}$ for each class $c = 1, \ldots, C$ are trained in order to fit to the observed label variables $y$ by a linear classification model on the topic activations. We make use of these weights $\eta$ by deriving a per topic thresholding according to learned importance for each topic: $\theta^{(i)} \geq \epsilon/\eta^{(i)}$.

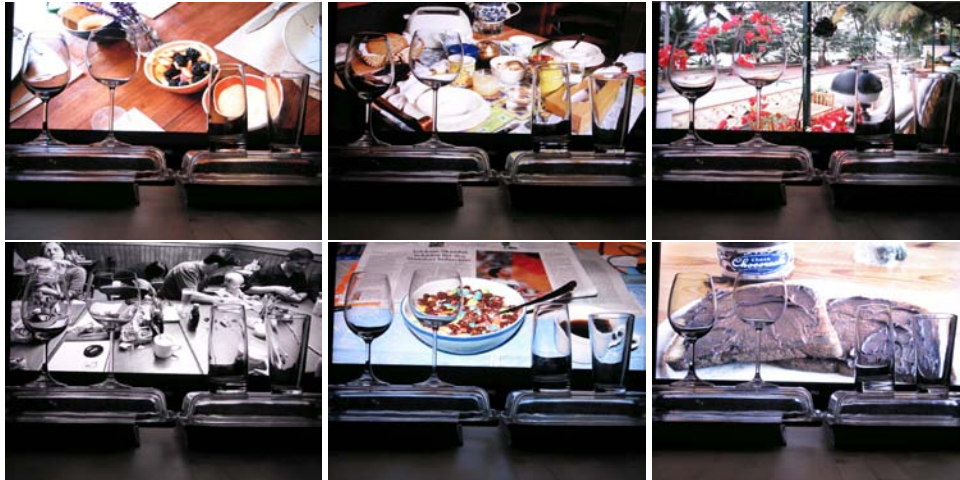

Figure 4: Example images from our training set of transparent objects in front of varying background.

## 4 Experiments

We have evaluated the proposed method on a glass detection task in a domestic environment under different view-point and illumination conditions; we compared to two baseline methods, HOG and vector quantized SIFT.

We collected data[2] for four glass objects (two wine glasses and two water glasses) in front of a LCD monitor with varying background (we used images from flickr.com under the search term 'breakfast table') in order to capture 200 training images of transparent objects. Figure 4 shows some example images of the training set.

We extracted a dense grid of 15 by 37 patches of each of the 800 glass examples as well as 800 background crops. Each patch is represented as a 4 by 4 grid of 9 dimensional edge orientation histograms. Neighboring patches overlap by $75\%$. We explored training the LDA model either only on foreground (glass) patches, only on background (non-transparent) patches, or on both, as reported below. The prior parameters for the LDA model were set to be $\alpha = 2$ and $\beta = 0.01$ and a total of 25 components were estimated. The components learnt from foreground patches are shown in Figure 2; patches from background or mixed data were qualitatively similar.

We infer latent topic activations for each patch and set detections of transparent visual words according to the above-threshold topic dimensions. We set the threshold corresponding to an average activation of 2 latent components per patch on the training set. Based on these 15 by 37 grids of transparent visual word occurrences, we train a linear, binary SVM in order to classify glasses vs. background.

For detection we follow the same procedure to infer the latent topic activations. Figure 3 shows example detections of transparent visual words on an example test image. We run a scanning window algorithm to detect likely object locations, examining all spatial location in the test image, and a range of scales from 0.5 to 1.5 with respect to the training size, in increments of 0.1. In each window latent topic activations are inferred for all descriptors and classification by the linear SVM is performed on the resulting grid of transparent visual word occurrences. For inference we use the implementation of [4], that results in an averaged computation time of $8.4$ms per descriptor on a single core of an Intel Core2 2.3 Ghz machine. This is substantial but not prohibitive, as we can reuse computation by choosing an appropriate stride of our scanning technique.

We compare to 2 baseline methods: traditional visual words and the histogram of oriented gradients (HOG) detector [6]. Both baselines share the same detection approach - namely obtaining detections by applying a linear SVM classifier in sliding window approach - but are based on very different

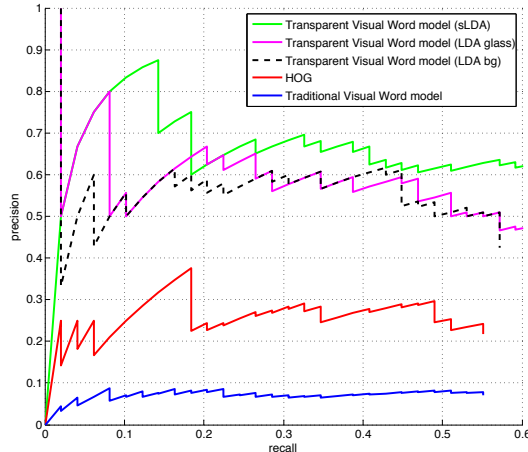

Figure 5: Performance evaluation of detector based on transparent visual words w.r.t. baseline. See text for details.

representations. For the traditional visual words baseline we replace the transparent visual words by visual words formed in a conventional fashion: sampled patches are directly input to a vector quantization scheme. We tried different number of clusters from 100 to 1000 and found $k = 100$ to work slightly better than the other choices. The HOG baseline basically leaves out any feature quantization and operates directly on the gradient histograms. We use the code provided by the authors [6].

To evaluate our approach, we recorded 14 test images of the above transparent objects in a home environment containing 49 glass instances in total; note that this test set is very different in nature from the training data. The training images were all collected with background illumination patterns obtained entirely from online image sources whereas the test data is under natural home illumination conditions. Further the training images were collected from a single viewpoint while viewpoint varies in the test data. In order to quantify our detection results we use the evaluation metric proposed in [8] with a matching threshold of $0.3$.

Our methods based on transparent visual words outperform both baselines across all ranges of operating points as shown in the precision-recall curve in Figure 5. We show results for the LDA model trained only on glass patches (LDA glass) as well as trained only on background patches (LDA bg). While neither of the methods achieve performance that would indicate glass detection is a solved problem, the results point in a promising direction. Example detections of our system on the test data are shown in Figure 6.

We also evaluated the supervised LDA as described above on data with mixed foreground and background patches, where the class label for each patch was provided to the training regime. The performance of sLDA is also displayed in Figure 5. In all of our experiments the transparent visual word models outperformed the conventional appearance baselines. Remarkably, latent topics learned on background data performed nearly as well as those trained on foreground data; those learned using a discriminative paradigm tended to outperform those trained in an unsupervised fashion, but the difference was not dramatic. Further investigation is needed to determine when discriminative models may have significant value, and/or whether a single latent representation is sufficient for a broad range of category recognition tasks.

## 5 Conclusion and Future Work

We have shown how appearance descriptors defined with an additive local factor model can capture local structure of transparent objects. Structures which are only weakly present in individual training instances can be revealed in a local factor model and inferred in test images. Learned latent topics define our "transparent visual words"; multiple such words can be detected at a single location. Recognition is performed using a conventional discriminative method and we show results for

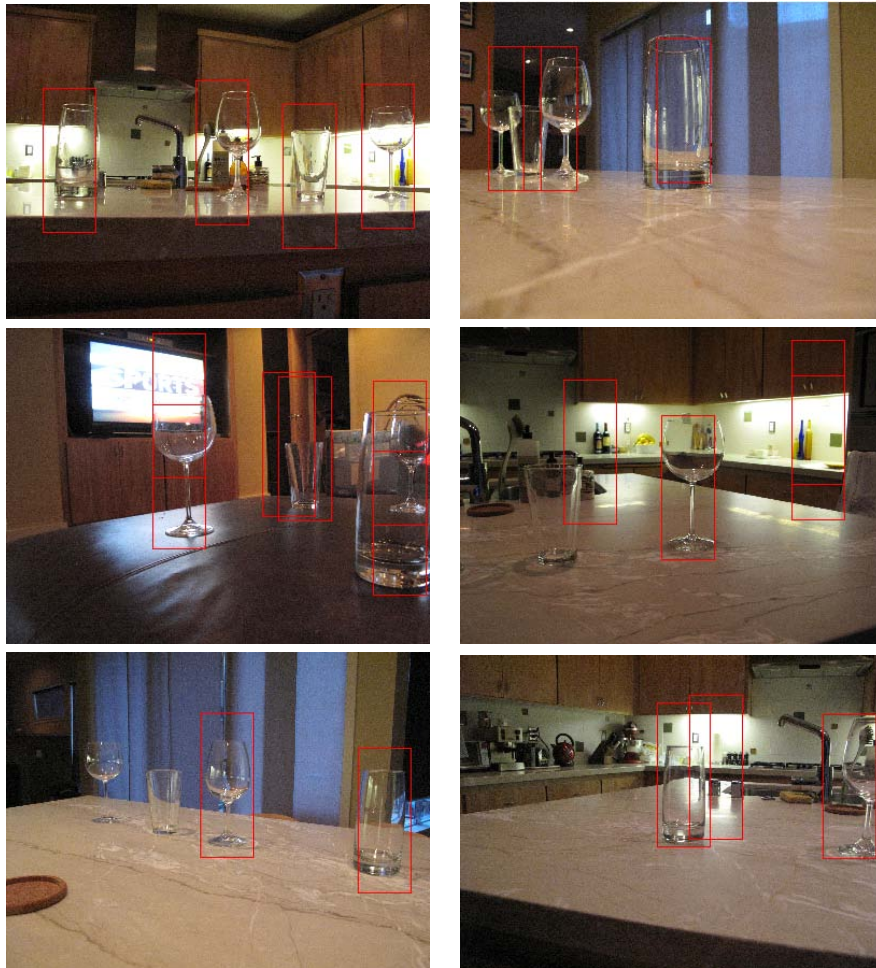

Figure 6: Example of transparent object detection with transparent local features.

detection of transparent glasses in a domestic environment. These results support our claim that an additive model of local patch appearance can be advantageous when modeling transparent objects, and that latent topic models such as LDA are appropriate for discovering locally transparent "visual words". This also demonstrates the advantage of estimating a latent appearance representation prior to a vector quantization step, in contrast to the conventional current approach of doing so in reverse.

We see this work as a first step toward transparent object recognition in complex environments. Our evaluation establishes a first baseline for transparent object recognition. While limited in scope, the range of test objects, poses and environments considered are varied and natural (i.e. not a laboratory environment). More extensive evaluation of these methods is needed with a wider range of poses, with more objects, occlusion and more varied illumination conditions.

There are several avenues of potential future work. We have not explicitly addressed specularity, which is often indicative of local shape, though specular features may be captured in our representation. Dense sampling may be suboptimal and it would be valuable to explore invariant detection schemes in the context of this overall method. Finally, we assume no knowledge of background statistics at test time, which may be overly restrictive; inferred background statistics may be informative in determining whether observed local appearance statistics are discriminative for a particular object category.

**Acknowledgements.** This work was supported in part by Willow Garage, Google, NSF grants IIS-0905647 and IIS-0819984, and a Feodor Lynen Fellowship granted by the Alexander von Humboldt Foundation.

## Footnotes

[1]our implementation is based on the one of [33]

[2]all data is available at `http://www.eecs.berkeley.edu/~mfritz/transparency`

# References

[1] E. H. Adelson. On seeing stuff: the perception of materials by humans and machines. In *SPIE*, 2001.

[2] A. C. Berg, T. L. Berg, and J. Malik. Shape matching and object recognition using low distortion correspondence. In *CVPR*, pages 26–33, 2005.

[3] D. Blei and J. McAuliffe. Supervised topic models. In *NIPS*, 2007.

[4] D. Blei, A. Ng, and M. Jordan. Latent dirichlet allocation. *JMLR*, 2003.

[5] G. Csurka, C. Dance, L. Fan, J. Willarnowski, and C. Bray. Visual categorization with bags of keypoints. In *SLCV'04*, pages 59–74, Prague, Czech Republic, May 2004.

[6] N. Dalal and B. Triggs. Histograms of oriented gradients for human detection. In *CVPR*, 2005.

[7] A. DelPozo and S. Savarese. Detecting specular surfaces on natural images. In *CVPR*, 2007.

[8] M. Everingham, A. Zisserman, C. K. I. Williams, and L. Van Gool. The PASCAL Visual Object Classes Challenge 2005 (VOC2005) Results. http://www.pascal-network.org/challenges/VOC/voc2005/results.pdf, 2005.

[9] P. F. Felzenszwalb, D. McAllester, and D. Ramana. A discriminatively trained, multiscale, deformable part model. In *CVPR*, 2008.

[10] R. Fergus, L. Fei-Fei, P. Perona, and A. Zisserman. Learning object categories from google's image search. In *ICCV*, 2005.

[11] R. Fergus, A. Zisserman, and P. Perona. Object class recognition by unsupervised scale-invariant learning. In *CVPR*, 2003.

[12] R. W. Fleming and H. H. Bülthoff. Low-level image cues in the perception of translucent materials. *ACM Trans. Appl. Percept.*, 2(3):346–382, 2005.

[13] M. Fritz and B. Schiele. Decomposition, discovery and detection of visual categories using topic models. In *CVPR*, 2008.

[14] K. Grauman and T. Darrell. The pyramid match kernel: Efficient learning with sets of features. *JMLR*, 8:725–760, 2007.

[15] T. L. Griffiths and M. Steyvers. Finding scientific topics. *PNAS USA*, 2004.

[16] T. Hofmann. Unsupervised learning by probabilistic latent semantic analysis. *Machine Learning*, 2001.

[17] S. Lazebnik, C. Schmid, and J. Ponce. Beyond bags of features: Spatial pyramid matching for recognizing natural scene categories. In *CVPR*, pages 2169–2178, 2006.

[18] D. Lowe. Distinctive image features from scale-invariant keypoints. *IJCV*, 60(2):91–110, 2004.

[19] J. Matas, O. Chum, U. Martin, and T. Pajdla. Robust wide baseline stereo from maximally stable extremal regions. In P. L. Rosin and A. D. Marshall, editors, *BMVC*, pages 384–393, 2002.

[20] K. McHenry and J. Ponce. A geodesic active contour framework for finding glass. In *CVPR*, 2006.

[21] K. McHenry, J. Ponce, and D. A. Forsyth. Finding glass. In *CVPR*, 2005.

[22] J. C. Niebles, H. Wang, and L. Fei-Fei. Unsupervised learning of human action categories using spatial-temporal words. *Int. J. Comput. Vision*, 79(3):299–318, 2008.

[23] P. Nillius and J.-O. Eklundh. Classifying materials from their reflectance properties. In T. Pajdla and J. Matas, editors, *ECCV*, volume 3021, 2004.

[24] D. Nister and H. Stewenius. Scalable recognition with a vocabulary tree. In *CVPR*, 2006.

[25] M. Osadchy, D. Jacobs, and R. Ramamoorthi. Using specularities for recognition. In *ICCV*, 2003.

[26] A. Quattoni, M. Collins, and T. Darrell. Conditional random fields for object recognition. In *NIPS*, 2004.

[27] M. Riesenhuber and T. Poggio. Hierarchical models of object recognition in cortex. *Nature Neuroscience*, 2:1019–1025, 1999.

[28] S. Roth and M. J. Black. Specular flow and the recovery of surface structure. In *CVPR*, 2006.

[29] E. Shechtman and M. Irani. Matching local self-similarities across images and videos. In *CVPR*, 2007.

[30] J. Sivic, B. C. Russell, A. A. Efros, A. Zisserman, and W. T. Freeman. Discovering objects and their locations in images. In *ICCV*, 2005.

[31] J. Sivic and A. Zisserman. Video Google: A text retrieval approach to object matching in videos. In *ICCV*, 2003.

[32] E. Sudderth, A. Torralba, W. Freeman, and A. Willsky. Learning hierarchical models of scenes, objects, and parts. In *ICCV*, 2005.

[33] C. Wang, D. Blei, and L. Fei-Fei. Simultaneous image classification and annotation. In *CVPR*, 2009.

[34] D. Zongker, D. Werner, B. Curless, and D. Salesin. Environment matting and compositing. In *SIGGRAPH*, pages 205–214, 1999.

